# Expectation Consistent Free Energies for Approximate Inference

**Manfred Opper**
ISIS
School of Electronics and
Computer Science
University of Southampton
SO17 1BJ, United Kingdom
`mo@ecs.soton.ac.uk`

**Ole Winther**
Informatics and
Mathematical Modelling
Technical University of Denmark
DK-2800 Lyngby, Denmark
`owi@imm.dtu.dk`

## Abstract

We propose a novel a framework for deriving approximations for intractable probabilistic models. This framework is based on a free energy (negative log marginal likelihood) and can be seen as a generalization of adaptive TAP [1, 2, 3] and expectation propagation (EP) [4, 5]. The free energy is constructed from two approximating distributions which encode different aspects of the intractable model such a single node constraints and couplings and are by construction consistent on a chosen set of moments. We test the framework on a difficult benchmark problem with binary variables on fully connected graphs and 2D grid graphs. We find good performance using sets of moments which either specify factorized nodes or a spanning tree on the nodes (structured approximation). Surprisingly, the Bethe approximation gives very inferior results even on grids.

## 1   Introduction

The development of tractable approximations for the statistical inference with probabilistic data models is of central importance in order to develop their full potential. The most prominent and widely developed [6] approximation technique is the so called *Variational Approximation* (VA) in which the true intractable probability distribution is approximated by the closest one in a tractable family. The most important tractable families of distributions are multivariate Gaussians and distributions which factorize in all or in certain groups of variables [7]. Both choices have their drawbacks. While factorizing distributions neglect correlations, multivariate Gaussians allow to retain a significant amount of dependencies but are restricted to continuous random variables which have the entire real space as their natural domain (otherwise KL divergences becomes infinite).

More recently a variety of non variational approximations have been developed which can be understood from the idea of global *consistency* between *local* approximations. E.g., in the Bethe–Kikuchi approach [8] the local neighborhood of each variable in a graphical model is implicitly approximated by a tree-like structure. Consistency is achieved by the matching of marginal distributions at the connecting edges of the graph. Thomas Minka's

*Expectation Propagation* (EP) framework seems to provide a general framework for developing and unifying such consistency approximations [4, 5]. Although the new frameworks have led to a variety of promising applications, often outperforming VA schemes, the unsatisfactory division between the treatment of constrained and unconstrained, continuous random variables seems to persist.

In this paper we propose an alternative approach which we call the *expectation consistent* (EC) approximation which is not plagued by this problem. We require consistency between *two* complimentary global approximations (say, a factorizing & a Gaussian one) to the same probabilistic model which may have different support. Our method is a generalization of the *adaptive TAP* approach (ADATAP) [2, 3] developed for inference on densely connected graphical models which has been applied successfully to a variety of problems ranging from probabilistic ICA over Gaussian process models to bootstrap methods for kernel machines.

## 2  Approximative inference

We consider the problem of computing expectations, i.e. certain sums or integrals involving a probability distribution with density

$$p(\mathbf{x}) = \frac{1}{Z}f(\mathbf{x}) \, , \tag{1}$$

for a vector of random variables $\mathbf{x} = (x_1, x_2, \ldots, x_N)$ with the *partition function* $Z = \int d\mathbf{x} f(\mathbf{x})$. We assume that the necessary exact operations are intractable, where the *intractability* arises either because the necessary sums are over a too large number of variables or because multivariate integrals cannot be evaluated exactly. In a typical scenario, $f(\mathbf{x})$ is expressed as a product of two functions

$$f(\mathbf{x}) = f_1(\mathbf{x})f_2(\mathbf{x}) \tag{2}$$

with $f_{1,2}(\mathbf{x}) \geq 0$, where $f_1$ is "simple" enough to allow for tractable computations. The idea of many approximate inference methods is to approximate the "complicated" part $f_2(\mathbf{x})$ by replacing it with a "simpler" function, say of some *exponential form* $\exp\left(\boldsymbol{\lambda}^T \mathbf{g}(\mathbf{x})\right) \equiv \exp\left(\sum_{j=1}^K \lambda_j g_j(\mathbf{x})\right)$. The vector of functions $\mathbf{g}$ is chosen in such a way that the desired sums or integrals can be calculated in an efficient way and the parameters $\boldsymbol{\lambda}$ are adjusted to optimize certain criteria. Hence, the word *tractability* should always be understood as relative to some approximating set of functions $\mathbf{g}$.

Our novel framework of approximation will be restricted to problems, where *both* parts $f_1$ and $f_2$ can be considered as tractable relative to some suitable $\mathbf{g}$, and the intractability of the density $p$ arises from forming their product. Take, as an example, the density (with respect to the Lebesgue measure in $R^N$) given by

$$p(\mathbf{x}) = \prod_\alpha \Psi_\alpha(\mathbf{x}_\alpha) \, \exp\left(\sum_{i<j} x_i J_{ij} x_j\right) \, , \tag{3}$$

where the $\mathbf{x}_\alpha$ denote tractable potentials defined on disjoint subsets of variables $\mathbf{x}_\alpha$. In order to have a non-trivial problem, the $\Psi_\alpha$ should be a *non-Gaussian* function. One may approximate $p(\mathbf{x})$ by a factorizing distribution, thereby replacing $f_2(\mathbf{x}) \equiv \exp\left(\sum_{i<j} x_i J_{ij} x_j\right)$ by some function which factorizes in the components $x_i$. Alternatively, one can consider replacing $f_1(\mathbf{x}) = \prod_i \Psi_i(x_i)$ by a Gaussian function to make the whole distribution *Gaussian*. Both approximations are not ideal. The first completely neglects correlations of the variables but leads to marginal distributions of the $x_i$, which might qualitatively resemble the non Gaussian shape of the true marginal. The second

one neglects the non Gaussian effects but incorporates correlations which might be used in order to approximate the two variable covariance functions. While within the VA both approximations would appear independent from each others, we will, in the following develop an approach for combining two approximations which "communicate" by matching the corresponding expectations of the functions $\mathbf{g}(\mathbf{x})$. We do *not* have to assume that either choice is a reasonably good approximation for the *global joint density* $p(\mathbf{x})$ as done in the VA. In fact, we apply the approach to a case where the KL divergence between one of them and $p$ is even *infinite*!

## 3 Gibbs free energy

Free energies (FE) provide a convenient formalism for dealing with probabilistic approximation problems. In this framework, the *true, intractable* distribution $p(\mathbf{x}) = \frac{f(\mathbf{x})}{Z}$ is *implicitly* characterized as the solution of an optimization problem defined through the the the relative entropy or KL divergence

$$KL(q,p) = \int d\mathbf{x}\, q(\mathbf{x}) \ln \frac{q(\mathbf{x})}{p(\mathbf{x})} \tag{4}$$

between $p$ and other trial distributions $q$. In contrast to the usual formulation of the VA, where one minimizes the KL divergence directly within a tractable family, it is more convenient to consider the following two stage optimization process. In the first step, one constrains the trial distributions $q$ by fixing the values of a set of generalized moments $\langle \mathbf{g}(\mathbf{x}) \rangle_q$. This will be helpful later to enable the communication between approximations. We define the Gibbs Free Energy $G(\boldsymbol{\mu})$ as

$$G(\boldsymbol{\mu}) = \min_q \left\{ KL(q,p) \mid \langle \mathbf{g}(\mathbf{x}) \rangle_q = \boldsymbol{\mu} \right\} - \ln Z \;. \tag{5}$$

We have subtracted the term $\ln Z$ to make the expression independent of the intractable partition function $Z$. In a second step, the moments of the distribution and also the partition function $Z$ are found within the same approach by relaxing the constraints and further minimizing $G(\boldsymbol{\mu})$ with respect to the $\boldsymbol{\mu}$.

$$\min_{\boldsymbol{\mu}} G(\boldsymbol{\mu}) = - \ln Z \qquad \text{and} \qquad \langle \mathbf{g} \rangle = \operatorname*{argmin}_{\boldsymbol{\mu}} G(\boldsymbol{\mu}) \;. \tag{6}$$

We will next give a short summary of properties of the Gibbs free energy (GFE). The optimizing density in (5) is given by

$$q(\mathbf{x}) = \frac{f(\mathbf{x})}{Z(\boldsymbol{\lambda})} \exp\left( \boldsymbol{\lambda}^T \mathbf{g}(\mathbf{x}) \right) \;, \tag{7}$$

with a normalizing partition function $Z(\boldsymbol{\lambda})$. The set of *Lagrange parameters* $\boldsymbol{\lambda} = \boldsymbol{\lambda}(\boldsymbol{\mu})$ is chosen such that the conditions $\langle \mathbf{g}(\mathbf{x}) \rangle_q = \boldsymbol{\mu}$ are fulfilled, i.e. $\boldsymbol{\lambda}$ satisfies

$$\frac{\partial \ln Z(\boldsymbol{\lambda})}{\partial \boldsymbol{\lambda}} = \boldsymbol{\mu} \;. \tag{8}$$

Inserting the optimizing distribution eq. (7) into the definition of the Gibbs free energy eq. (5), we get the explicit expression:

$$G(\boldsymbol{\mu}) = - \ln Z(\boldsymbol{\lambda}(\boldsymbol{\mu})) + \boldsymbol{\lambda}^T(\boldsymbol{\mu})\boldsymbol{\mu} = \max_{\boldsymbol{\lambda}} \left\{ - \ln Z(\boldsymbol{\lambda}) + \boldsymbol{\lambda}^T \boldsymbol{\mu} \right\} \;, \tag{9}$$

i.e., $G$ is the *Legendre transform* or *dual* of $- \ln Z(\boldsymbol{\lambda})$. Hence, $G$ is a convex function of its arguments and $\frac{\partial G(\boldsymbol{\mu})}{\partial \boldsymbol{\mu}} = \boldsymbol{\lambda}$.

$G(\boldsymbol{\mu})$ can be used to generate moments, e.g.

$$\frac{\partial^2 G(\boldsymbol{\mu})}{\partial \boldsymbol{\mu} \partial \boldsymbol{\mu}^T} = \frac{\partial \boldsymbol{\lambda}}{\partial \boldsymbol{\mu}^T} = \left(\frac{\partial \boldsymbol{\mu}^T}{\partial \boldsymbol{\lambda}}\right)^{-1} = \left[\langle \mathbf{g}(\mathbf{x})\mathbf{g}^T(\mathbf{x})\rangle - \langle \mathbf{g}(\mathbf{x})\rangle\langle \mathbf{g}(\mathbf{x})\rangle^T\right]^{-1} , \quad (10)$$

where the expectations are over the density eq. (7).

The derivative with respect to a parameter $t$ contained in the probability density $p(\mathbf{x}|t) = \frac{f(\mathbf{x},t)}{Z_t}$ can be calculated using (9) and (8) as

$$\frac{dG(\boldsymbol{\mu}, t)}{dt} = -\frac{\partial \ln Z(\boldsymbol{\lambda}, t)}{\partial t} + \left(\boldsymbol{\mu} - \frac{\partial \ln Z(\boldsymbol{\lambda}, t)}{\partial \boldsymbol{\lambda}}\right)\frac{d\boldsymbol{\lambda}^T}{dt} = -\frac{\partial \ln Z(\boldsymbol{\lambda}, t)}{\partial t} . \quad (11)$$

The important message is that we only need to take the explicit $t$ dependence into account, i.e. we can keep $\boldsymbol{\lambda}$ fixed upon differentiation.

### 3.1 Free energy examples

Here we give the free energies for three tractable models and choices of moments that will be used subsequently in the free energy framework.

**Completely factorized,** i.e. $p(\mathbf{x}) = \prod_i \psi_i(x_i)$. For simplicity we will consider biased binary variables: $\Psi_i(x_i) = [\delta(x_i+1)+\delta(x_i-1)]e^{\theta_i x_i}$ and fix the first moments $\mathbf{m} = \langle \mathbf{x}\rangle$. Denoting the conjugate Lagrange parameters by $\boldsymbol{\gamma}$:

$$G(\mathbf{m}) = \sum_i G_i(m_i) \qquad \text{with} \qquad G_i(m_i) = \max_{\gamma_i}\left\{-\ln Z_i(\gamma_i) + m_i\gamma_i\right\} \quad (12)$$

with $Z_i(\gamma_i) = \int dx_i \; \Psi_i(\xi)e^{\gamma_i x_i} = 2\cosh(\gamma_i + \theta_i)$.

**Tree-connected graph.** For the case where either the couplings and the moments together define a tree-connected graph, we can write the free energy in term of single- and two-node free energies. Considering again completely factorized binary variables, all non-trivial moments on the graph $(ij) \in \mathcal{G}$ are the means $\mathbf{m}$ and correlations of linked nodes $M_{ij} = \langle x_i x_j\rangle$:

$$G(\mathbf{m}, \{M_{ij}\}_{(ij)\in\mathcal{G}}) = \sum_{(ij)\in\mathcal{G}} G_{ij}(m_i, m_j, M_{ij}) + \sum_i (1 - n_i)G_i(m_i) , \quad (13)$$

where $G_{ij}(m_i, m_j, M_{ij})$ is the two-node free energy defined in a similar fashion as the one-node free energy, $n_i$ the number of links to node $i$ and $G_i(m_i)$ is the one-node free energy.

**Gaussian distribution.** We set $\boldsymbol{\mu} = (\mathbf{m}, \mathbf{M})$ with all first moments $\mathbf{m}$ and an arbitrary subset of second moments $\mathbf{M}$ for a Gaussian model $\Psi_i(x_i) \propto \exp[a_i x_i - \frac{b_i}{2}x_i^2]$ and $p(\mathbf{x})$ given by eq. (3). We introduce conjugate variables $\boldsymbol{\gamma}$ and $-\boldsymbol{\Lambda}/2$. $\boldsymbol{\gamma}$ can be eliminated analytically, whereas we get a log-determinant maximization problem for $\boldsymbol{\Lambda}$:

$$G(\mathbf{m}, \mathbf{M}) = -\frac{1}{2}\mathbf{m}^T\mathbf{J}\mathbf{m} - \mathbf{m}^T\mathbf{a} + \frac{1}{2}\sum_i M_{ii}b_i \quad (14)$$

$$+ \max_{\boldsymbol{\Lambda}}\left\{\frac{1}{2}\ln\det(\boldsymbol{\Lambda} - \mathbf{J}) - \frac{1}{2}\operatorname{Tr}\boldsymbol{\Lambda}(\mathbf{M} - \mathbf{m}\mathbf{m}^T)\right\} .$$

## 4 Exact interpolation representation

If the density $p$ factors into a tractable $f_1$ and an intractable part $f_2$, according to eq. (2), we can construct a representation of the Gibbs free energy which also separates into two corresponding parts. We treat $f_2(\mathbf{x})$ as a *perturbation* which is smoothly turned on using a parameter $0 \leq t \leq 1$. We define $f_2(\mathbf{x}, t)$ to be a smooth *interpolation* between the trivial $f_2(\mathbf{x}, t = 0) = 1$ and the "full" intractable $f_2(\mathbf{x}, t = 1) = f_2(\mathbf{x})$. Hence, we define parametric densities and the corresponding free energy by $p(\mathbf{x}|t) = \frac{1}{Z_t} f_1(\mathbf{x}) f_2(\mathbf{x}, t)$ and

$$q(\mathbf{x}|t) = \frac{1}{Z_q(\boldsymbol{\lambda}, t)} f_1(\mathbf{x}) f_2(\mathbf{x}, t) \exp\left(\boldsymbol{\lambda}^T \mathbf{g}(\mathbf{x})\right) \tag{15}$$

$$G_q(\boldsymbol{\mu}, t) = \max_{\boldsymbol{\lambda}} \left\{ -\ln Z_q(\boldsymbol{\lambda}, t) + \boldsymbol{\lambda}^T \boldsymbol{\mu} \right\} . \tag{16}$$

Using eq. (11), and the fact that $\frac{\partial \ln Z(\boldsymbol{\lambda}, t)}{\partial t} = \left\langle \frac{d \ln f_2(\mathbf{x}, t)}{dt} \right\rangle_{q(\mathbf{x}|t)}$ we derive the following exact identity for the free energy $G(\boldsymbol{\mu}, t)$

$$G_q(\boldsymbol{\mu}, 1) - G_q(\boldsymbol{\mu}, 0) = -\int_0^1 dt \left\langle \frac{d \ln f_2(\mathbf{x}, t)}{dt} \right\rangle_{q(\mathbf{x}|t)} . \tag{17}$$

to relate the Gibbs free energy of the intractable model $G_q(\boldsymbol{\mu}) = G(\boldsymbol{\mu}, t = 1)$ and tractable model $G(\boldsymbol{\mu}, t = 0)$. An simple approximation is obtained for the case $f_2(\mathbf{x}, t) = [f_2(\mathbf{x})]^t$, when the expectation over $q(\mathbf{x}|t)$ is replaced by the expectation over the tractable $q(\mathbf{x}|0)$, ie

$$G(\boldsymbol{\mu}) \approx G(\boldsymbol{\mu}, 0) - \int_0^1 dt \left\langle \frac{d \ln f_2(\mathbf{x}, t)}{dt} \right\rangle_{q(\mathbf{x}|0)} = G(\boldsymbol{\mu}, 0) - \langle \ln f_2(\mathbf{x}) \rangle_{q(\mathbf{x}|0)} . \tag{18}$$

This result coincides with the variational approximation when we restrict the family of optimizing functions to be of the form $q(\mathbf{x}|0)$.

### 4.1 Expectation Consistent Approximation

Our goal is to go beyond the variational approximation and capture more of the $t$ dependence of the intractable part $f_2$ in the term eq. (17). We will now use our assumption that besides the family of distributions eq. (15), there is a second family which can be used as an approximation to the distribution $p(\mathbf{x}|t)$. It is given by

$$r(\mathbf{x}|t) = \frac{1}{Z_r(\boldsymbol{\lambda}, t)} f_2(\mathbf{x}, t) \exp\left(\boldsymbol{\lambda}^T \mathbf{g}(\mathbf{x})\right) , \tag{19}$$

where the parameters $\boldsymbol{\lambda}$ will be chosen in such a way as to guarantee *consistency for the expectations* of $\mathbf{g}$, i.e. $\langle \mathbf{g}(\mathbf{x}) \rangle_{r(\mathbf{x}|t)} = \boldsymbol{\mu}$. Using eq. (19) in place of $q(\mathbf{x}|t)$ in eq. (17), we get the approximation

$$G_q(\boldsymbol{\mu}, 1) - G_q(\boldsymbol{\mu}, 0) \approx \int_0^1 dt \left\langle \frac{d \ln f_2(\mathbf{x}, t)}{dt} \right\rangle_{r(\mathbf{x}|t)} = G_r(\boldsymbol{\mu}, 1) - G_r(\boldsymbol{\mu}, 0) , \tag{20}$$

where the last equality is derived from the fact that both types of densities eqs. (15) and (19) contain the same exponential family. This allows us to carry out the integral over the interaction strength $t$ in eq. (20) in closed form without specifying the interpolating term $f_2(\mathbf{x}, t)$ explicitly. Hence, the *expectation consistent (EC)* approximation is

$$G_q(\boldsymbol{\mu}, 1) \approx G_q(\boldsymbol{\mu}, 0) + G_r(\boldsymbol{\mu}, 1) - G_r(\boldsymbol{\mu}, 0) \equiv G^{\mathrm{EC}}(\boldsymbol{\mu}) . \tag{21}$$

To simplify notation, we will write $G_q \equiv G_q(\boldsymbol{\mu}, 0)$, $G_r \equiv G_r(\boldsymbol{\mu}, 1)$ and $G_s \equiv G_r(\boldsymbol{\mu}, 0)$ in the following.

## 5 Models with pair-wise interactions

Our framework works very naturally to the class of models eq. (3). The EC approximation eq. (21) will be based on approximating neglected correlations in the factorizing approximation using a *Gaussian* distributions. The corresponding free energies $G(\boldsymbol{\mu}, 1)$ and $r(\boldsymbol{\mu}, 0)$ appearing in eq. (21) are thus found using the Gaussian free energy eq. (14) with $\mathbf{J}$ and $\mathbf{J} = \mathbf{0}$

$$
\begin{aligned}
G^{\mathrm{EC}}(\mathbf{m}, \mathbf{M}) \;=\; & G_q(\mathbf{m}, \mathbf{M}, 0) - \frac{1}{2}\mathbf{m}^T \mathbf{J} \mathbf{m} \\
& + \max_{\boldsymbol{\Lambda}} \left\{ \frac{1}{2} \ln \det(\boldsymbol{\Lambda} - \mathbf{J}) - \frac{1}{2} \operatorname{Tr} \boldsymbol{\Lambda}(\mathbf{M} - \mathbf{m}\mathbf{m}^T) \right\} \\
& - \max_{\boldsymbol{\Lambda}} \left\{ \frac{1}{2} \ln \det \boldsymbol{\Lambda} - \frac{1}{2} \operatorname{Tr} \boldsymbol{\Lambda}(\mathbf{M} - \mathbf{m}\mathbf{m}^T) \right\} \;,
\end{aligned}
\tag{22}
$$

where the free energy $G_q(\mathbf{m}, \mathbf{M}, 0)$ will depend explicitly upon the potentials $\Psi_\alpha(\mathbf{x}_\alpha)$.

## 6 Free energy minimization algorithms

In our approach, inference is based on the minimization of the free energy with respect to its arguments $\boldsymbol{\mu}$. While the exact free energy is by construction a convex function in $\boldsymbol{\mu}$, our free energy *approximation* $G_{\mathrm{EC}} = G_q + G_r - G_s$ contains the concave contribution $-G_s$ and may not be convex. Hence, we may have potentially many local minima and other stationary points, like saddlepoints. Moreover, the expression derived for $G_{\mathrm{EC}}$ is not a simple explicit functions of $\boldsymbol{\mu}$ but contains additional optimizations over Lagrange parameters. Consequently, we cannot expect that *message-passing* algorithms similar to *loopy belief propagation* or *expectation propagation (EP)* [4] which sequentially update moments and Lagrange parameters are guaranteed to converge.

### 6.1 Guaranteed convergence – variational bounding

An iterative algorithms which is guaranteed to find at least stationary points of the Free Energy is similar in spirit to the so-called double-loop approaches [9, 10]. The basic idea is to minimize a decreasing sequence of *convex upper bounds* to $G_{\mathrm{EC}}$. Each convex upper bound is derived by linearizing the concave term $-G_s(\boldsymbol{\mu})$ at the present iteration $\boldsymbol{\mu}^*$, i.e. using $G_s(\boldsymbol{\mu}) \geq G_s^{\mathrm{lbound}}(\boldsymbol{\mu}) = -C_* + \boldsymbol{\mu}^T \boldsymbol{\lambda}_s^*$, with $C_* \equiv \ln Z_q(\boldsymbol{\lambda}_s^*)$ and $\boldsymbol{\lambda}_s^* = \boldsymbol{\lambda}_s(\boldsymbol{\mu}^*)$. Since it is usually much easier to deal with the Lagrange parameters we convert the resulting convex minimization problem into a *concave maximization problem* for the $\boldsymbol{\lambda}$'s

$$
\begin{aligned}
G_{\mathrm{EC}}(\boldsymbol{\mu}) \;\leq\; & G_q(\boldsymbol{\mu}) + G_r(\boldsymbol{\mu}) - \boldsymbol{\mu}^T \boldsymbol{\lambda}_s^* + C_* \\
=\; & \min_{\boldsymbol{\mu}} \max_{\boldsymbol{\lambda}_q, \boldsymbol{\lambda}_r} \left\{ -\ln Z_q(\boldsymbol{\lambda}_q) - \ln Z_r(\boldsymbol{\lambda}_r) + \boldsymbol{\mu}^T (\boldsymbol{\lambda}_q + \boldsymbol{\lambda}_r - \boldsymbol{\lambda}_s^*) + C_* \right\} \\
=\; & \max_{\boldsymbol{\lambda}_q, \boldsymbol{\lambda}_r} \left\{ -\ln Z_q(\boldsymbol{\lambda}_q) - \ln Z_r(\boldsymbol{\lambda}_r) | \boldsymbol{\lambda}_q + \boldsymbol{\lambda}_r = \boldsymbol{\lambda}_s^* \right\} + C_* \\
=\; & \max_{\boldsymbol{\lambda}_r} \left\{ -\ln Z_q(\boldsymbol{\lambda}_s^* - \boldsymbol{\lambda}_r) - \ln Z_r(\boldsymbol{\lambda}_r) + C_* \right\} \;.
\end{aligned}
\tag{23}
$$

This can be summarized in the following double loop algorithm which is guaranteed not to increase our free energy approximation.

1. Outer loop: For fixed old value $\boldsymbol{\mu}^*$, bound the concave term $-G_s(\boldsymbol{\mu})$ by $-G_s^{\mathrm{lbound}}(\boldsymbol{\mu})$ go get the convex upper bound to $G_{\mathrm{EC}}(\boldsymbol{\mu})$.

2. Inner loop: Solve the concave maximization problem

$$
\max_{\boldsymbol{\lambda}_r} \mathcal{L} \qquad \text{with} \qquad \mathcal{L} = -\ln Z_q(\boldsymbol{\lambda}_s^* - \boldsymbol{\lambda}_r) - \ln Z_r(\boldsymbol{\lambda}_r) \;.
\tag{24}
$$

Inserting the solution into $\boldsymbol{\mu}(\boldsymbol{\lambda}_r) = \langle \mathbf{g}(\mathbf{x}) \rangle_r$ gives the new value $\boldsymbol{\mu}^*$ for $\boldsymbol{\mu}$.

Currently, we either solve the non-linear inner-loop optimization by a sequential approach that are computationally efficient when $G_r$ is the free energy of a multivariate Gaussian or by interior point methods [11, 12].

## 7 Simulations

We have tested the EC framework in a benchmark set-up proposed by Wainwright and Jordan [12]. The stochastic variables are binary $x_i = \pm 1$ with pair-wise couplings are used. The $N = 16$ nodes are either fully connected or connected to nearest neighbors in a 4-by-4 grid. The external field (observation) strengths $\theta_i$ are drawn from a *uniform* distribution $\theta_i \sim \mathcal{U}[-d_{\mathrm{obs}}, d_{\mathrm{obs}}]$ with $d_{\mathrm{obs}} = 0.25$. Three types of coupling strength statistics are considered: repulsive (anti-ferromagnetic) $J_{ij} \sim \mathcal{U}[-2d_{\mathrm{coup}}, 0]$, mixed $J_{ij} \sim \mathcal{U}[-d_{\mathrm{coup}}, +d_{\mathrm{coup}}]$ and attractive (ferromagnetic) $J_{ij} \sim \mathcal{U}[0, +2d_{\mathrm{coup}}]$ with $d_{\mathrm{coup}} > 0$. We compute the average one-norm error on the marginals: $\sum_i |p(\mathbf{x}_i = 1) - p(\mathbf{x}_i = 1|\mathrm{Method})|/N$, $p(\mathbf{x}_i = 1) = (1 + m_i)/2$ over 100 trials testing the following Methods: SP = sum-product (aka loopy belief propagation (BP) or Bethe approximation) and LD = log-determinant maximization [12], EC factorized and EC structured. Results for SP and LD are taken from Ref. [12]. For EC, we are minimizing the EC free energy eq. (22) where $G_q(\mathbf{m}, \mathbf{M}, 0)$ depend upon the approximation we are using. For the factorized model we use the free energy eq. (12) and for the structured model we assume a *single* tractable potential $\psi(\mathbf{x})$ in eq. (3) which contains all couplings on a spanning tree. For $G_q$, we use the free energy eq. (13). The spanning tree is defined by the following simple heuristic: choose as next pair of nodes to link, the (so far unlinked) pair with strongest absolute coupling $|J_{ij}|$ that will not cause a loop in the graph.

The results are summarized in table 1. The Bethe approximation always give inferior results compared to EC (note that only loopy BP convergent problem instances were used to calculate the error [12]). This might be a bit surprising for the sparsely connected grids. This indicates that loopy BP and too a lesser degree extensions building upon BP [5] are only to be applied to really sparse graphs and/or weakly coupled nodes, where the error induced by not using a properly normalized distribution can be expected to be small. We also speculate that a structured variational approximation, using the same heuristics as described above to construct the spanning tree, in many cases will be superior to the Bethe approximation as also observed by Ref. [5]. LD is a robust method which seems to be limited in it's achievable precision. EC structured is uniformly superior to all other approaches. Additional simulations (not included in the paper) also indicate that EC give much improved estimates of free energies and two-node marginals when compared to the Bethe- and Kikuchi-approximation.

## 8 Conclusion and outlook

We have introduced a novel method for approximate inference which tries to overcome certain limitations of single approximating distributions by achieving consistency for two of these on the same problem. While we have demonstrated its accuracy in this paper only for a model with binary elements, it can also be applied to models with continuous random variables or hybrid models with both discrete and continuous variables. We expect that our method becomes most powerful when certain tractable substructures of variables with strong dependencies can be identified in a model. Our approach would then allow to deal well with the weaker dependencies between the groups. A generalization of our method to treat graphical models beyond pair-wise interaction is obtained by iterating the approximation. This is useful in cases, where an initial three term approximation $G^{EC} =$

Table 1: The average one-norm error on marginals for the Wainwright-Jordan set-up.

| Problem type | | | Method | | | |
|---|---|---|---|---|---|---|
| | | | SP | LD | EC fac | EC struct |
| Graph | Coupling | $d_{\mathrm{coup}}$ | Mean | Mean | Mean | Mean |
| Full | Repulsive | 0.25 | 0.037 | 0.020 | 0.003 | 0.0017 |
| | Repulsive | 0.50 | 0.071 | 0.018 | 0.031 | 0.0143 |
| | Mixed | 0.25 | 0.004 | 0.020 | 0.002 | 0.0013 |
| | Mixed | 0.50 | 0.055 | 0.021 | 0.022 | 0.0151 |
| | Attractive | 0.06 | 0.024 | 0.027 | 0.004 | 0.0031 |
| | Attractive | 0.12 | 0.435 | 0.033 | 0.117 | 0.0211 |
| Grid | Repulsive | 1.0 | 0.294 | 0.047 | 0.153 | 0.0031 |
| | Repulsive | 2.0 | 0.342 | 0.041 | 0.198 | 0.0021 |
| | Mixed | 1.0 | 0.014 | 0.016 | 0.011 | 0.0018 |
| | Mixed | 2.0 | 0.095 | 0.038 | 0.082 | 0.0068 |
| | Attractive | 1.0 | 0.440 | 0.047 | 0.125 | 0.0028 |
| | Attractive | 2.0 | 0.520 | 0.042 | 0.177 | 0.0024 |

$G_q + G_r - G_s$ still contains non-tractable component free energies $G$.

## References

[1] M. Opper and O. Winther, "Gaussian processes for classification: Mean field algorithms," *Neural Computation*, vol. 12, pp. 2655–2684, 2000.

[2] M. Opper and O. Winther, "Tractable approximations for probabilistic models: The adaptive Thouless-Anderson-Palmer mean field approach," *Phys. Rev. Lett.*, vol. 86, pp. 3695, 2001.

[3] M. Opper and O. Winther, "Adaptive and self-averaging Thouless-Anderson-Palmer mean field theory for probabilistic modeling," *Phys. Rev. E*, vol. 64, pp. 056131, 2001.

[4] T. P. Minka, "Expectation propagation for approximate Bayesian inference," in *UAI 2001*, 2001, pp. 362–369.

[5] T. Minka and Y. Qi, "Tree-structured approximations by expectation propagation," in *NIPS 16*, S. Thrun, L. Saul, and B. Schölkopf, Eds. MIT Press, Cambridge, MA, 2004.

[6] Christopher M. Bishop, David Spiegelhalter, and John Winn, "Vibes: A variational inference engine for bayesian networks," in *Advances in Neural Information Processing Systems 15*, S. Thrun S. Becker and K. Obermayer, Eds., pp. 777–784. MIT Press, Cambridge, MA, 2003.

[7] H. Attias, "A variational Bayesian framework for graphical models," in *Advances in Neural Information Processing Systems 12*, T. Leen et al., Ed. 2000, MIT Press, Cambridge.

[8] J. S. Yedidia, W. T. Freeman, and Y. Weiss, "Generalized belief propagation," in *Advances in Neural Information Processing Systems 13*, T. K. Leen, T. G. Dietterich, and V. Tresp, Eds., 2001, pp. 689–695.

[9] A. L. Yuille, "CCCP algorithms to minimize the Bethe and Kikuchi free energies: convergent alternatives to belief propagation," *Neural Comput.*, vol. 14, no. 7, pp. 1691–1722, 2002.

[10] T. Heskes, K. Albers, and H. Kappen, "Approximate inference and constrained optimization," in *UAI-03*, San Francisco, CA, 2003, pp. 313–320, Morgan Kaufmann Publishers.

[11] S. Boyd and L. Vandenberghe, *Convex Optimization*, Cambridge University Press, 2004.

[12] M. J. Wainwright and M. I. Jordan, "Semidefinite methods for approximate inference on graphs with cycles," Tech. Rep. UCB/CSD-03-1226, UC Berkeley CS Division, 2003.
